# Whose Vote Should Count More: Optimal Integration of Labels from Labelers of Unknown Expertise

**Jacob Whitehill, Paul Ruvolo, Tingfan Wu, Jacob Bergsma, and Javier Movellan**
Machine Perception Laboratory
University of California, San Diego
La Jolla, CA, USA
{ jake, paul, ting, jbergsma, movellan }@mplab.ucsd.edu

## Abstract

Modern machine learning-based approaches to computer vision require very large databases of hand labeled images. Some contemporary vision systems already require on the order of millions of images for training (e.g., Omron face detector [9]). New Internet-based services allow for a large number of labelers to collaborate around the world at very low cost. However, using these services brings interesting theoretical and practical challenges: (1) The labelers may have wide ranging levels of expertise which are unknown a priori, and in some cases may be adversarial; (2) images may vary in their level of difficulty; and (3) multiple labels for the same image must be combined to provide an estimate of the actual label of the image. Probabilistic approaches provide a principled way to approach these problems. In this paper we present a probabilistic model and use it to simultaneously infer the label of each image, the expertise of each labeler, and the difficulty of each image. On both simulated and real data, we demonstrate that the model outperforms the commonly used "Majority Vote" heuristic for inferring image labels, and is robust to both noisy and adversarial labelers.

## 1   Introduction

In recent years machine learning-based approaches to computer vision have helped to greatly accelerate progress in the field. However, it is now becoming clear that many practical applications require very large databases of hand labeled images. The labeling of very large datasets is becoming a bottleneck for progress. One approach to address this incoming problem is to make use of the vast human resources on the Internet. Indeed, projects like the ESP game [17], the Listen game[16], Soylent Grid [15], and reCAPTCHA [18] have revealed the possibility of harnessing human resources to solve difficult machine learning problems. While these approaches use clever schemes to obtain data from humans for free, a more direct approach is to hire labelers online. Recent Web tools such as Amazon's *Mechanical Turk* [1] provide ideal solutions for high-speed, low cost labeling of massive databases.

Due to the distributed and anonymous nature of these tools, interesting theoretical and practical challenges arise. For example, principled methods are needed to combine the labels from multiple experts and to estimate the certainty of the current labels. Which image should be labeled (or relabeled) next must also be decided – it may be prudent, for example, to collect many labels for each image in order to increase one's confidence in that image's label. However, if an image is easy and the labelers of that image are reliable, a few labels may be sufficient and valuable resources may be used to label other images. In practice, combining the labels of multiple coders is a challenging process due to the fact that: (1) The labelers may have wide ranging levels of expertise which are

unknown a priori, and in some cases may be adversarial; (2) images may also vary in their level of difficulty, in a manner that may also be unknown a priori.

Probabilistic methods provide a principled way to approach this problem using standard inference tools. We explore one such approach by formulating a probabilistic model of the labeling process, which we call *GLAD* (Generative model of Labels, Abilities, and Difficulties), and using inference methods to simultaneously infer the expertise of each labeler, the difficulty of each image, and the most probable label for each image. On both simulated and real-life data, we demonstrate that the model outperforms the commonly used "Majority Vote" heuristic for inferring image labels, and is robust to both adversarial and noisy labelers.

## 2 Modeling the Labeling Process

Consider a database of $n$ images, each of which belongs to one of two possible categories of interest (e.g., face/non-face; male/female; smile/non-smile; etc.). We wish to determine the class label $Z_j$ (0 or 1) of each image $j$ by querying from $m$ labelers. The observed labels depend on several causal factors: (1) the difficulty of the image; (2) the expertise of the labeler; and (3) the true label. We model the difficulty of image $j$ using the parameter $1/\beta_j \in [0, \infty)$ where $\beta_j$ is constrained to be positive. Here $1/\beta_j = \infty$ means the image is very ambiguous and hence even the most proficient labeler has a 50% chance of labeling it correctly. $1/\beta_j = 0$ means the image is so easy that even the most obtuse labeler will always label it correctly.

The expertise of each labeler $i$ is modeled by the parameter $\alpha_i \in (-\infty, +\infty)$. Here an $\alpha = +\infty$ means the labeler always labels images correctly; $-\infty$ means the labeler always labels the images *in*correctly, i.e., he/she can distinguish between the two classes perfectly but always inverts the label, either maliciously or because of a consistent misunderstanding. In this case ($\alpha_i < 0$), the labeler is said to be *adversarial*. Finally, $\alpha_i = 0$ means that the labeler cannot discriminate between the two classes – his/her labels carry no information about the true image label $Z_j$. Note that we do not require the labelers to be human – labelers can also be, for instance, automatic classifiers. Hence, the proposed approach will provide a principled way of combining labels from any combination of human and previously existing machine-based classifiers.

The labels given by labeler $i$ to image $j$ (which we call the *given labels*) are denoted as $L_{ij}$ and, under the model, are generated as follows:

$$p(L_{ij} = Z_j | \alpha_i, \beta_j) = \frac{1}{1 + e^{-\alpha_i \beta_j}} \tag{1}$$

Thus, under the model, the log odds for the obtained labels being correct are a bilinear function function of the difficulty of the label and the expertise of the labeler, i.e.,

$$\log \frac{p(L_{ij} = Z_j)}{1 - p(L_{ij} = Z_j)} = \alpha_i \beta_j \tag{2}$$

More skilled labelers (higher $\alpha_i$) have a higher probability of labeling correctly. As the difficulty $1/\beta_j$ of an image increases, the probability of the label being correct moves toward 0.5. Similarly, as the labeler's expertise decreases (lower $\alpha_i$), the chance of correctness likewise drops to 0.5. Adversarial labelers are simply labelers with negative $\alpha$.

Figure 1 shows the causal structure of the model. True image labels $Z_j$, labeler accuracy values $\alpha_i$, and image difficulty values $\beta_j$ are sampled from a known prior distribution. These determine the observed labels according to Equation 1. Given a set of observed labels $\mathbf{l} = \{l_{ij}\}$, the task is to infer simultaneously the most likely values of $\mathbf{Z} = \{Z_j\}$ (the true image labels) as well as the labeler accuracies $\boldsymbol{\alpha} = \{\alpha_i\}$ and the image difficulty parameters $\boldsymbol{\beta} = \{\beta_j\}$. In the next section we derive the Maximum Likelihood algorithm for inferring these values.

## 3 Inference

The observed labels are samples from the $\{L_{ij}\}$ random variables. The unobserved variables are the true image labels $Z_j$, the different labeler accuracies $\alpha_i$, and the image difficulty parameters $1/\beta_j$. Our goal is to efficiently search for the most probable values of the unobservable variables

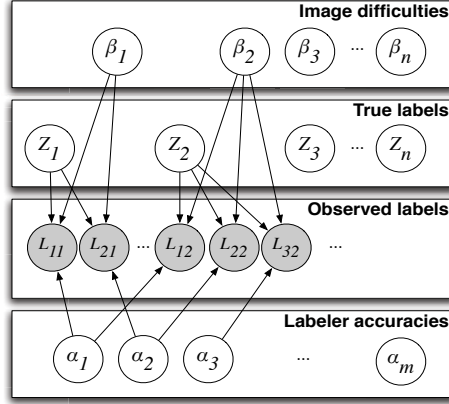

Figure 1: Graphical model of image difficulties, true image labels, observed labels, and labeler accuracies. Only the shaded variables are observed.

$\mathbf{Z}$, $\boldsymbol{\alpha}$ and $\boldsymbol{\beta}$ given the observed data. Here we can use Expectation-Maximization approach (EM) to obtain maximum likelihood estimates of the parameters of interest (the full derivation is in the Supplementary Materials):

**E step**: Let the set of all given labels for an image $j$ be denoted as $\mathbf{l}_j = \{l_{ij'} \mid j' = j\}$. Note that not every labeler must label every single image. In this case, the index variable $i$ in $l_{ij'}$ refers only to those labelers who labeled image $j$. We need to compute the posterior probabilities of all $z_j \in \{0, 1\}$ given the $\boldsymbol{\alpha}, \boldsymbol{\beta}$ values from the last M step and the observed labels:

$$
\begin{aligned}
p(z_j|\mathbf{l}, \boldsymbol{\alpha}, \boldsymbol{\beta}) &= p(z_j|\mathbf{l}_j, \boldsymbol{\alpha}, \beta_j) \\
&\propto p(z_j|\boldsymbol{\alpha}, \beta_j)p(\mathbf{l}_j|z_j, \boldsymbol{\alpha}, \beta_j) \\
&\propto p(z_j)\prod_i p(l_{ij}|z_j, \alpha_i, \beta_j)
\end{aligned}
$$

where we noted that $p(z_j|\boldsymbol{\alpha}, \beta_j) = p(z_j)$ using the conditional independence assumptions from the graphical model.

**M step**: We maximize the standard auxiliary function $Q$, which is defined as the expectation of the joint log-likelihood of the observed and hidden variables $(\mathbf{l}, \mathbf{Z})$ given the parameters $(\boldsymbol{\alpha}, \boldsymbol{\beta})$, w.r.t. the posterior probabilities of the $\mathbf{Z}$ values computed during the last E step:

$$
\begin{aligned}
Q(\boldsymbol{\alpha}, \boldsymbol{\beta}) &= E\left[\ln p(\mathbf{l}, \mathbf{z}|\boldsymbol{\alpha}, \boldsymbol{\beta})\right] \\
&= E\left[\ln \prod_j \left(p(z_j)\prod_i p(l_{ij}|z_j, \alpha_i, \beta_j)\right)\right] \\
&\quad \text{since } l_{ij} \text{ are cond. indep. given } \mathbf{z}, \boldsymbol{\alpha}, \boldsymbol{\beta} \\
&= \sum_j E\left[\ln p(z_j)\right] + \sum_{ij} E\left[\ln p(l_{ij}|z_j, \alpha_i, \beta_j)\right]
\end{aligned}
$$

where the expectation is taken over $\mathbf{z}$ given the old parameter values $\boldsymbol{\alpha}^{old}, \boldsymbol{\beta}^{old}$ as estimated during the last E-step. Using gradient ascent, we find values of $\boldsymbol{\alpha}$ and $\boldsymbol{\beta}$ that locally maximize $Q$.

### 3.1 Priors on $\alpha, \beta$

The $Q$ function can be modified straightforwardly to handle a prior over each $\alpha_i$ and $\beta_j$ by adding a log-prior term for each of these variables. These priors may be useful, for example, if we know that most labelers are not adversarial. In this case, the prior for $\alpha$ can be made very low for $\alpha < 0$.

The prior probabilities are also useful when the ground-truth $Z$ value of particular images is (somehow) known for certain. By "clamping" the $Z$ values (using the prior) for the images on which the

true label is known for sure, the model may be able to better estimate the other parameters. The $Z$ values for such images can be clamped by setting the prior probability $p(z_j)$ (used in the E-Step) for these images to be very high towards one particular class. In our implementation we used Gaussian priors ($\mu = 1, \sigma = 1$) for $\alpha$. For $\beta$, we need a prior that does not generate negative values. To do so we re-parameterized $\beta \doteq e^{\beta'}$ and imposed a Gaussian prior ($\mu = 1, \sigma = 1$) on $\beta'$.

## 3.2 Computational Complexity

The computational complexity of the E-Step is linear in the number of images and the total number of labels. For the M-Step, the values of $Q$ and $\nabla Q$ must be computed repeatedly until convergence.[1] Computing each function is linear in the number of images, number of labelers, and total number of image labels.

Empirically when using the approach on a database of 1 million images that we recently collected and labeled we found that the EM procedure converged in about 10 minutes using a single core of a Xeon 2.8 GHz processor. The algorithm is parallelizable and hence this running time could be reduced substantially using multiple cores. Real time inference may also be possible if we maintain parameters close to the solution that are updated as new labels become available. This would allow using the algorithm in an active manner to choose in real-time which images should be labeled next so as to minimize the uncertainty about the image labels.

## 4 Simulations

Here we explore the performance of the model using a set of image labels generated by the model itself. Since, in this case we know the parameters $\mathbf{Z}, \alpha$, and $\beta$ that generated the observed labels, we can compare them with corresponding parameters estimated using the EM procedure.

In particular, we simulated between 4 and 20 labelers, each labeling 2000 images, whose true labels $\mathbf{Z}$ were either 0 or 1 with equal probability. The accuracy $\alpha_i$ of each labeler was drawn from a normal distribution with mean 1 and variance 1. The inverse-difficulty for each image $\beta_j$ was generated by exponentiating a draw from a normal distribution with mean 1 and variance 1. Given these labeler abilities and image difficulties, the observed labels $l_{ij}$ were sampled according to Equation 1 using $\mathbf{Z}$. Finally, the EM inference procedure described above was executed to estimate $\alpha, \beta, \mathbf{Z}$. This procedure was repeated 40 times to smooth out variability between trials. On each trial we computed the correlation between the parameter estimates $\hat{\alpha}, \hat{\beta}$ and the true parameter values $\alpha, \beta$. The results (averaged over all 40 experimental runs) are shown in Figure 2. As expected, as the number of labelers grows, the parameter estimates converge to the true values.

We also computed the proportion of label estimates $\hat{\mathbf{Z}}$ that matched the true image labels $\mathbf{Z}$. We compared the maximum likelihood estimates of the GLAD model to estimates obtained by taking the majority vote as the predicted label. The predictions of the proposed GLAD model were obtained by thresholding at 0.5 the posterior probability of the label of each image being of class 1 given the accuracy and difficulty parameters returned by EM (see Section 3). Results are shown in Figure 2. GLAD makes fewer errors than the majority vote heuristic. The difference between the two approaches is particularly pronounced when the number of labelers per image is small. On many images, GLAD correctly infers the true image label $\mathbf{Z}$ even when that $\mathbf{Z}$ value was the *minority* opinion. In essence, GLAD is exploiting the fact that some labelers are experts (which it infers automatically), and hence their votes should count more on these images than the votes of less skilled labelers.

**Modeling Image Difficulty** : To explore the importance of estimating image difficulty we performed a simple simulation: Image labels (0 or 1) were assigned randomly (with equal probability) to 1000 images. Half of the images were "hard", and half were "easy." Fifty simulated labelers labeled all 1000 images. The proportion of "good" to "bad" labelers is 25:1. The probability of correctness for each image difficulty and labeler quality combination was given by the table below:

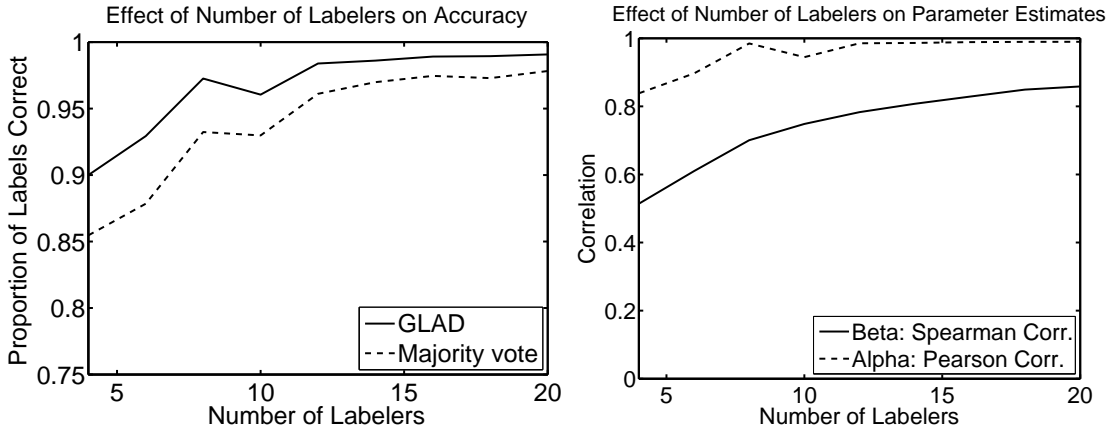

Figure 2: **Left:** The accuracies of the GLAD model versus simple voting for inferring the underlying class labels on simulation data. **Right:** The ability of GLAD to recover the true alpha and beta parameters on simulation data.

|  | Image Type | |
|---|---|---|
| **Labeler type** | Hard | Easy |
| Good | 0.95 | 1 |
| Bad | 0.54 | 1 |

We measured performance in terms of proportion of correctly estimated labels. We compared three approaches: (1) our proposed method, GLAD; (2) the method proposed in [5], which models labeler ability but not image difficulty; and (3) Majority Vote. The simulations were repeated 20 times and average performance calculated for the three methods. The results shown below indicated that modeling image difficulty can result in significant performance improvements.

| Method | Error |
|---|---|
| GLAD | 4.5% |
| Majority Vote | 11.2% |
| Dawid & Skene [5] | 8.4% |

### 4.1 Stability of EM under Various Starting Points

Empirically we found that the EM procedure was fairly insensitive to varying the starting point of the parameter values. In a simulation study of 2000 images and 20 labelers, we randomly selected each $\alpha_i \sim U[0,4]$ and $\log(\beta_j) \sim U[0,3]$, and EM was run until convergence. Over the 50 simulation runs, the average percent-correct of the inferred labels was $85.74\%$, and the standard deviation of the percent-correct over all the trials was only $0.024\%$.

## 5   Empirical Study I: Greebles

As a first test-bed for GLAD using real data obtained from the Mechanical Turk, we posted pictures of 100 "Greebles" [6], which are synthetically generated images that were originally created to study human perceptual expertise. Greebles somewhat resemble human faces and have a "gender": Males have horn-like organs that point up, whereas for females the horns point down. See Figure 3 (left) for examples. Each of the 100 Greeble images was labeled by 10 different human coders on the Turk for gender (male/female). Four greebles of each gender (separate from the 100 labeled images) were given as examples of each class. Shown at a resolution of 48x48 pixels, the task required careful inspection of the images in order to label them correctly. The ground-truth gender values were all known with certainty (since they are rendered objects) and thus provided a means of measuring the accuracy of inferred image labels.

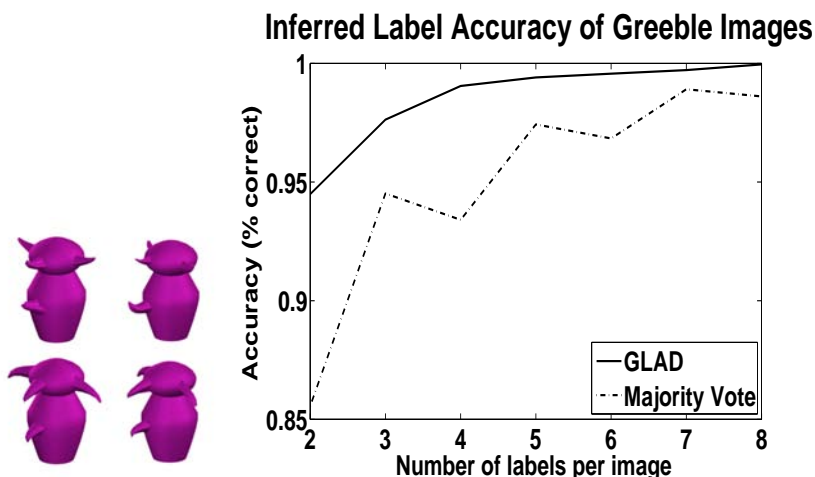

Figure 3: **Left**: Examples of Greebles. The top two are "male" and the bottom two are "female."
**Right**: Accuracy of the inferred labels, as a function of the number of labels $M$ obtained for each
image, of the Greeble images using either GLAD or Majority Vote. Results were averaged over 100
experimental runs.

We studied the effect of varying the number of labels $M$ obtained from different labelers for each
image, on the accuracy of the inferred $\mathbf{Z}$. Hence, from the 10 labels total we obtained per Greeble
image, we randomly sampled $2 \leq M \leq 8$ labels over all labelers during each experimental trial. On
each trial we compared the accuracy of labels $\mathbf{Z}$ as estimated by GLAD (using a threshold of $0.5$
on $p(Z)$) to labels as estimated by the Majority Vote heuristic. For each value of $M$ we averaged
performance for each method over 100 trials.

Results are shown in Figure 3 (right). For all values of $M$ we tested, the labels as inferred by GLAD
are significantly higher than for Majority Vote ($p < 0.01$). This means that, in order to achieve the
same level of accuracy, fewer labels are needed. Moreover, the variance in accuracy was less for
GLAD than for Majority Vote for all $M$ that were tested, suggesting that the quality of GLAD's
outputs is more stable than of the heuristic method. Finally, notice how, for the even values of $M$,
the Majority Vote accuracy decreases. This may stem from the lack of optimal decision rule under
Majority Vote when an equal number of labelers say an image is Male as who say it is Female.
GLAD, since it makes its decisions by also taking ability and difficulty into account, does not suffer
from this problem.

## 6  Empirical Study II: Duchenne Smiles

As a second experiment, we used the Mechanical Turk to label face images containing smiles as
either *Duchenne* or *Non-Duchenne*. A Duchenne smile ("enjoyment" smile) is distinguished from a
Non-Duchenne ("social" smile) through the activation of the *Orbicularis Oculi* muscle around the
eyes, which the former exhibits and the latter does not (see Figure 4 for examples). Distinguishing
the two kinds of smiles has applications in various domains including psychology experiments,
human-computer interaction, and marketing research. Reliable coding of Duchenne smiles is a
difficult task even for certified experts in the Facial Action Coding System.

We obtained Duchenne/Non-Duchenne labels for 160 images from 20 different Mechanical Turk
labelers; in total, there were 3572 labels. (Hence, labelers labeled each image a variable number of
times.) For ground truth, these images were also labeled by two certified experts in the Facial Action
Coding System. According to the expert labels, 58 out of 160 images contained Duchenne smiles.

Using the labels obtained from the Mechanical Turk, we inferred the image labels using either
GLAD or the Majority Vote heuristic, and then compared them to ground truth.

**Duchenne Smiles**                                                             **Non-Duchenne Smiles**

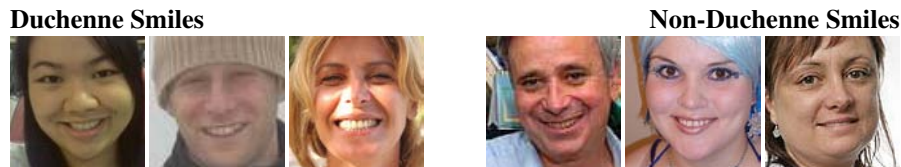

Figure 4: Examples of Duchenne (left) and Non-Duchenne (right) smiles. The distinction lies in the activation of *Orbicularis Oculi* muscle around the eyes, and is difficult to discriminate even for experts.

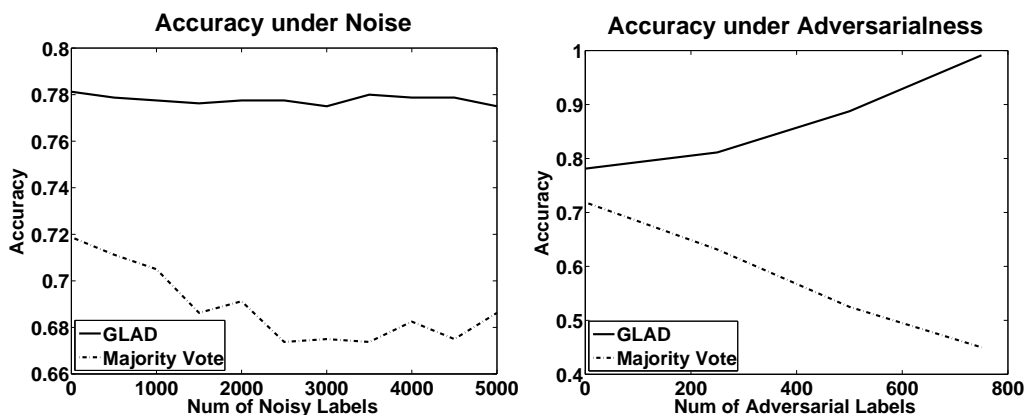

Figure 5: Accuracy (percent correct) of inferred Duchenne/Non-Duchenne labels using either GLAD or Majority Vote under (**left**) noisy labelers or (**right**) adversarial labelers. As the number of noise/adversarial labels increases, the performance of labels inferred using Majority Vote decreases. GLAD, in contrast, is robust to these conditions.

**Results**: Using just the raw labels obtained from the Mechanical Turk, the labels inferred using GLAD matched the ground-truth labels on $78.12\%$ of the images, whereas labels inferred using Majority Vote were only $71.88\%$ accurate. Hence, GLAD resulted in about a $6\%$ performance gain.

**Simulated Noisy and Adversarial Labelers**: We also simulated noisy and adversarial labeler conditions. It is to be expected, for example, that in some cases labelers may just try to complete the task in a minimum amount of time disregarding accuracy. In other cases labelers may misunderstand the instructions, or may be adversarial, thus producing labels that tend to be opposite to the true labels. Robustness to such noisy and adversarial labelers is important, especially as the popularity of Web-based labeling tools increases, and the quality of labelers becomes more diverse. To investigate the robustness of the proposed approaches we generated data from virtual "labelers" whose labels were completely uninformative, i.e., uniformly random. We also added artificial "adversarial" labelers whose labels tended to be the *opposite* of the true label for each image.

The number of noisy labels was varied from 0 to 5000 (in increments of 500), and the number of adversarial labels was varied from 0 to 750 (in increments of 250). For each setting, label inference accuracy was computed for both GLAD and the Majority Vote method. As shown in Figure 5, the accuracy of GLAD-based label inference is much less affected from labeling noise than is Majority Vote. When adversarial labels are introduced, GLAD automatically inferred that some labelers were purposely giving the opposite label and automatically flipped their labels. The Majority Vote heuristic, in contrast, has no mechanism to recover from this condition, and the accuracy falls steeply.

# 7 Related Work

To our knowledge GLAD is the first model in the literature to simultaneously estimate the true label, item difficulty, and coder expertise in an unsupervised and efficient manner.

Our work is related to the literature on standardized tests, particularly the Item Response Theory (IRT) community (e.g., Rasch [10], Birnbaum [3]). The GLAD model we propose in this paper can be seen as an unsupervised version of previous IRT models for the case in which the correct answers (i.e., labels) are unknown.

Snow, et al [14] used a probabilistic model similar to Naive Bayes to show that by averaging multiple naive labelers ($<= 10$) one can obtain labels as accurate as a few expert labelers. Two key differences between their model and GLAD are that: (1) they assume a significant proportion of images have been pre-labeled with ground truth values, and (2) all the images have equal difficulty. As we show in this paper, modeling image difficulty may be very important in some cases. Sheng, et al [12] examine how to identify which images of an image dataset to label again in order to reduce uncertainty in the posterior probabilities of latent class labels.

Dawid and Skene [5] developed a method to handle polytomous latent class variables. In their case the notion of "ability" is handled using full confusion matrices for each labeler. Smyth, et al [13] used a similar approach to combine labels from multiple experts for items with homogeneous levels of difficulty. Batchelder and Romney [2] infer test answers and test-takers' abilities simultaneously, but do not estimate item difficulties and do not admit adversarial labelers.

Other approaches employ a Bayesian model of the labeling process that considers both variability in labeler accuracies as well as item difficulty (e.g. [8, 7, 11]). However, inference in these models is based on MCMC which is likely to suffer from high computational expense, and the need to wait (arbitrarily long) for parameters to "burn in" during sampling.

# 8 Summary and Further Research

An important bottleneck facing the machine learning community is the need for very large datasets with hand-labeled data. Datasets whose scale was unthinkable a few years ago are becoming commonplace today. The Internet makes it possible for people around the world to cooperate on the labeling of these datasets. However, this makes it unrealistic for individual researchers to obtain the ground truth of each label with absolute certainty. Algorithms are needed to automatically estimate the reliability of ad-hoc anonymous labelers, the difficulty of the different items in the dataset, and the probability of the true labels given the currently available data.

We proposed one such system, GLAD, based on standard probabilistic inference on a model of the labeling process. The approach can handle the millions of parameters (one difficulty parameter per image, and one expertise parameter per labeler) needed to process large datasets, at little computational cost. The model can be used seamlessly to combine labels from both human labelers and automatic classifiers. Experiments show that GLAD can recover the true data labels more accurately than the Majority Vote heuristic, and that it is highly robust to both noisy and adversarial labelers.

**Active Sampling**: One advantage of probabilistic models is that they lend themselves to implementing active methods (e.g., Infomax [4]) for selecting which images should be re-labeled next. We are currently pursuing the development of control policies for optimally choosing whether to obtain more labels for a particular item – so that the inferred $Z$ label for that item becomes more certain – versus obtaining more labels from a particular *labeler* – so that his/her accuracy $\alpha$ may be better estimated, and *all* the images that he/she labeled can have their posterior probability estimates of $Z$ improved.

A software implementation of GLAD is available at `http://mplab.ucsd.edu/~jake`.

## Footnotes

[1]The `libgsl` conjugate gradient descent optimizer we used requires both $Q$ and $\nabla Q$.

# References

[1] Amazon. Mechanical turk. `http://www.mturk.com`.

[2] W. H. Batchelder and A. K. Romney. Test theory without an answer key. *Psychometrika*, 53(1):71–92, 1988.

[3] A. Birnbaum. Some latent trait models and their use in inferring an examinee's ability. *Statistical theories of mental test scores*, 1968.

[4] N. Butko and J. Movellan. I-POMDP: An infomax model of eye movement. In *Proceedings of the International Conference on Development and Learning*, 2008.

[5] A. Dawid and A. Skene. Maximum likelihood estimation of observer error-rates using the em algorithm. *Applied Statistics*, 28(1):20–28, 1979.

[6] I. Gauthier and M. Tarr. Becoming a "greeble" expert: Exploring mechanisms for face recognition. *Vision Research*, 37(12), 1997.

[7] V. Johnson. On bayesian analysis of multi-rater ordinal data: An application to automated essay grading. *Journal of the American Statistical Association*, 91:42–51, 1996.

[8] G. Karabatsos and W. H. Batchelder. Markov chain estimation for test theory without an answer key. *Psychometrika*, 68(3):373–389, 2003.

[9] Omron. OKAO vision brochure, July 2008.

[10] G. Rasch. *Probabilistic Models for Some Intelligence and Attainment Tests*. Denmark, 1960.

[11] S. Rogers, M. Girolami, and T. Polajnar. Semi-parametric analysis of multi-rater data. *Statistics and Computing*, 2009.

[12] V. Sheng, F. Provost, and P. Ipeirotis. Get another label? improving data quality and data mining using multiple noisy labelers. In *Knowledge Discovery and Data Mining*, 2008.

[13] P. Smyth, U. Fayyad, M. Burl, P. Perona, and P. Baldi. Inferring ground truth from subjective labelling of venus images. In *Advances of Neural Information Processing Systems*, 1994.

[14] R. Snow, B. O'Connor, D. Jurafsky, and A. Y. Ng. Cheap and fast - but is it good? evaluating non-expert annotations for natural language tasks. In *Proceedings of the 2008 Conference on Empirical Methods on Natural Language Processing*, 2008.

[15] S. Steinbach, V. Rabaud, and S. Belongie. Soylent grid: it's made of people! In *International Conference on Computer Vision*, 2007.

[16] D. Turnbull, R. Liu, L. Barrington, and G. Lanckriet. A Game-based Approach for Collecting Semantic Annotations of Music. In *8th International Conference on Music Information Retrieval (ISMIR)*, 2007.

[17] L. von Ahn and L. Dabbish. Labeling Images with A Computer Game. In *Proceedings of the SIGCHI conference on Human factors in computing systems*, pages 319–326. ACM Press New York, NY, USA, 2004.

[18] L. von Ahn, B. Maurer, C. McMillen, D. Abraham, and M. Blum. reCAPTCHA: Human-Based Character Recognition via Web Security Measures. *Science*, 321(5895):1465, 2008.

